# Generative versus discriminative training of RBMs for classification of fMRI images

**Tanya Schmah**

Department of Computer Science
University of Toronto
Toronto, Canada
schmah@cs.toronto.edu

**Geoffrey E. Hinton**

Department of Computer Science
University of Toronto
Toronto, Canada
hinton@cs.toronto.edu

**Richard S. Zemel**

Department of Computer Science
University of Toronto
Toronto, Canada
zemel@cs.toronto.edu

**Steven L. Small**

Department of Neurology
The University of Chicago
Chicago, USA
small@uchicago.edu

**Stephen Strother**

The Rotman Research Institute Baycrest
Toronto, Canada
sstrother@rotman-baycrest.on.ca

## Abstract

Neuroimaging datasets often have a very large number of voxels and a very small number of training cases, which means that overfitting of models for this data can become a very serious problem. Working with a set of fMRI images from a study on stroke recovery, we consider a classification task for which logistic regression performs poorly, even when L1- or L2- regularized. We show that much better discrimination can be achieved by fitting a generative model to each separate condition and then seeing which model is most likely to have generated the data. We compare discriminative training of exactly the same set of models, and we also consider convex blends of generative and discriminative training.

## 1 Introduction

Pattern classification approaches to analyzing functional neuroimaging data have become increasingly popular [12] [3] [4]. These approaches allow one to use well-founded classification methods to test whether the imaging data contains enough information to discriminate between different conditions. They may also lead to insight into underlying neural representations, highlighting brain regions that are most informative with respect to particular experimental variables.

One difficulty in applying these models is the paucity of data: the number of images available to analyze is typically very small relative to the data dimensionality, particularly if one does not want to restrict *a priori* the input to subsets of the voxels. Generative models are therefore of great interest, because building a density model of the imaging data itself can often uncover features of the data that are useful for classification as well as for generation. In regimes in which the number of training examples is relatively small, it has been shown that classifiers based on generative models can outperform discriminative classifiers, e.g., naive Bayes classifiers can beat logistic regression [11].

In this paper we investigate ways of using generative models to improve the discrimination of different conditions in functional neuroimaging data. Our primary interest with respect to the imaging data is to elucidate the brain changes that occur during recovery from a stroke. Towards this aim, we define an early-late discrimination task to see if the learning approach can find properties that distinguish pre-recovery from post-recovery scans.

## 2 Restricted Boltzmann Machines

A set of fMRI volumes can be modeled using a two-layer network called a "Restricted Boltzmann Machine" (RBM) [5], in which stochastic "visible units" are connected to stochastic "hidden units" using symmetrically weighted connections. The visible units of the RBM correspond to voxels, while the hidden units can be thought of as feature detectors. In the typical RBM, both visible and hidden units are binary, but we use a version in which the visible units are continuous and have Gaussian marginal distributions [15] [7] [1]. For simplicity, and since we are free to scale the data, we choose unit variance for the marginal distributions of the visible units.

The *energy* of the joint configuration $(\mathbf{v}, \mathbf{h})$ of the visible and hidden units is

$$E(\mathbf{v}, \mathbf{h}) := -\sum_{i,j} v_i w_{ij} h_j - \sum_j c_j h_j + \frac{1}{2} \sum_i (v_i - b_i)^2, \tag{1}$$

where $w_{ij}, b_i, c_j$ are fixed parameters. The joint distribution over visible and hidden variables is

$$P(\mathbf{v}, \mathbf{h}) := \frac{1}{Z} \exp\left(-E(\mathbf{v}, \mathbf{h})\right), \tag{2}$$

with *partition function* $Z := \int d\mathbf{u} \sum_{\mathbf{g}} \exp\left(-E(\mathbf{u}, \mathbf{g})\right)$.

The marginal distribution over the visible units can be expressed as:

$$P(\mathbf{v}) = \sum_{\mathbf{h}} P(\mathbf{v}, \mathbf{h}) = \frac{1}{Z} \exp\left(-F(\mathbf{v})\right), \tag{3}$$

where $F$ is the *free energy*:

$$F(\mathbf{v}) = -\log\left(\sum_{\mathbf{h}} \exp\left(-E(\mathbf{v}, \mathbf{h})\right)\right)$$

$$= -\sum_j \log\left(1 + \exp\left(v_i w_{ij} + c_j\right)\right) + \frac{1}{2} \sum_i (v_i - b_i)^2. \tag{4}$$

The marginal distribution over the visible units is typically intractable because of the partition function $Z$. However Gibbs sampling can be used to sample from an approximation to the marginal distribution, since the conditional probability distributions $P(\mathbf{v}|\mathbf{h})$ and $P(\mathbf{h}|\mathbf{v})$ are tractable:

$$P(\mathbf{v}|\mathbf{h}) = \prod_i \mathcal{N}\left(b_i + \sum_j w_{ij} h_j, \, 1\right), \qquad P(\mathbf{h}|\mathbf{v}) = \prod_j \sigma\left(\sum_i v_i w_{ij} + c_j\right),$$

where $\sigma$ is the logistic function, $\sigma(z) := 1/1 + \exp(-z)$. Note that the conditional probabilities of the hidden units are the same as for binary-only RBMs.

The aim of generative training of an RBM is to model the marginal distribution of the visible units $P(\mathbf{v})$. In maximum likelihood learning, the aim is to minimize the negative log probability of the training data,

$$L_{\text{gen}} = -\sum_{\mathbf{v} \in \mathcal{S}} \log P(\mathbf{v}|\theta), \tag{5}$$

where $\mathcal{S}$ is the training set and $\theta$ is the vector of all parameters $w_{ij}, b_i, c_j$. The gradient of this function is intractable, however there is an approximation to maximum likelihood learning called

Contrastive Divergence (CD), which works well in practice [5]. We use a $n$-step version of CD, with $n$ equal to either 3 or 6. At each iteration, the parameter increments are:

$$\Delta w_{ij} = \langle v_i h_j \rangle_0 - \langle v_i h_j \rangle_n$$
$$\Delta b_i = \langle v_i - b_i \rangle_0 - \langle v_i - b_i \rangle_n$$
$$\Delta c_j = \langle h_j \rangle_0 - \langle h_j \rangle_n$$

In this definition, angle brackets denote expected value over a certain distribution over the visible units, with the hidden units distributed according to the conditional distribution $P(\mathbf{h}|\mathbf{v})$. A subscript 0 indicates that the data distribution is used, i.e., visible units are given values corresponding to observed fMRI volumes; while a subscript $n$ indicates that $n$ steps of Gibbs sampling have been done, beginning at data points, to give an approximation to an expected value over the true distribution $P(\mathbf{v})$.

## 3   Classification using multiple RBMs

We consider binary classification tasks. The methods clearly generalize to arbitrary numbers of classes.

### 3.1   Classification via (mostly) generative training

We begin by generatively training two independent RBMs, one for each data class. For maximum likelihood learning, the cost function is the negative log probability of the training data:

$$L_{\text{gen}} = L_{\text{gen}}^A(\mathcal{S}_A) + L_{\text{gen}}^B(\mathcal{S}_B) := - \sum_{\mathbf{v} \in \mathcal{S}_A} \log P(\mathbf{v}|\theta_A) - \sum_{\mathbf{v} \in \mathcal{S}_B} \log P(\mathbf{v}|\theta_B), \qquad (6)$$

where $\mathcal{S}_A$ and $\mathcal{S}_B$ are the training data from classes $A$ and $B$, and $\theta_A$ and $\theta_B$ are the parameter vectors for the two RBMs. In practice we regularize by adding a term to this cost function that corresponds to putting a prior distribution on the weights $w_{ij}$.

In general, given probabilistic generative models for each of two classes, A and B, data can be classified by Bayes' theorem. For brevity, we write "A" for "$\mathbf{v}$ is of class A", and similarly for B. If we assume that $\mathbf{v}$ is *a priori* equally likely to belong to both classes, then

$$P(A|\mathbf{v}) = \frac{P(\mathbf{v}|A)}{P(\mathbf{v}|A) + P(\mathbf{v}|B)}.$$

If the distributions $P(\mathbf{v}|A)$ and $P(\mathbf{v}|B)$ are defined by RBMs, then they can be expressed in terms of free energies $F_A$ and $F_B$ and partition functions $Z_A$ and $Z_B$, as in Equation (3). Substituting this into Bayes' theorem gives

$$P(A|\mathbf{v}) = \sigma \left( F_B(\mathbf{v}) - F_A(\mathbf{v}) - T \right), \qquad (7)$$

where $T := \log (Z_A/Z_B)$. The free energies in this formula can be calculated easily using (4). However the partition functions are intractable for RBMs with large numbers of hidden and visible units. For this reason, we replace the unknown "threshold" $T$ with an independent parameter $\Delta$, and fit it *discriminatively*. (Thus this method is not *pure* generative training.) The aim of discriminative training is to model the conditional probability of the class labels given the visible units. In maximum likelihood learning, the cost function to be minimized is the negative log conditional probability of the class labels of the training data,

$$L_{\text{disc}} = - \sum_{\mathbf{v} \in \mathcal{S}} \log P(\text{class of } \mathbf{v}|\mathbf{v}, \theta_A, \theta_B, \Delta). \qquad (8)$$

### 3.2   Classification via discriminative training

As an alternative to generative training, the function $L_{disc}$ (defined in the previous equation) can be minimized directly, with respect to all parameters simultaneously: the $w_{ij}, b_i$ and $c_j$ of both RBMs and the threshold parameter $\Delta$. We use exactly the same model of $P(\text{class of } \mathbf{v}|\mathbf{v})$ as before,

summarized in Equation (7). By substituting Equations (7) and (4) into Equation (8) the gradient of $L_{disc}$ with respect to all parameters can be calculated exactly.

Substituting Equations (7) into Equation (8) gives

$$C = - \sum_{\mathbf{v} \in \mathcal{S}_A} \log \sigma \left( F_B(\mathbf{v}) - F_A(\mathbf{v}) - \Delta \right) - \sum_{\mathbf{v} \in \mathcal{S}_B} \log \sigma \left( \Delta + F_A(\mathbf{v}) - F_B(\mathbf{v}) \right).$$

where $\mathcal{S}_A$ and $\mathcal{S}_B$ are the sets of training data in classes $A$ and $B$.

Since $\frac{d}{dz} \log \sigma(z) = \sigma(-z)$, the partial derivative of $C$ with respect to the threshold parameter is:

$$\frac{\partial C}{\partial \Delta} = \sum_{\mathbf{v} \in \mathcal{S}_A} \sigma \left( \Delta + F_A(\mathbf{v}) - F_B(\mathbf{v}) \right) - \sum_{\mathbf{v} \in \mathcal{S}_B} \sigma \left( F_B(\mathbf{v}) - F_A(\mathbf{v}) - \Delta \right).$$

The free energies depend on the weights of the RBMs (suppressed in the above equation for ease of notation). If the parameters for the two RBMs are not linked in any way, then any given parameter $\theta$ affects either model A or model B but not both, so either $\partial F_B / \partial \theta = 0$ or $\partial F_A / \partial \theta = 0$. From (4) it follows that

$$\frac{\partial}{\partial w_{ij}} F(\mathbf{v}) = -p_j v_i, \quad \frac{\partial}{\partial c_j} F(\mathbf{v}) = -p_j, \quad \frac{\partial}{\partial b_i} F(\mathbf{v}) = b_i - v_i,$$

where $p_j := \sigma(z_j) = P(h_j|\mathbf{v})$. It follows that, setting $M(\mathbf{v}) := \sigma \left( F_B(\mathbf{v}) - F_A(\mathbf{v}) - \Delta \right)$, the derivatives for the parameters of model A are:

$$\frac{\partial C}{\partial w_{ij}} = \sum_{\mathbf{v} \in \mathcal{S}_A} (1 - M(\mathbf{v})) (-v_i \, p_j) + \sum_{\mathbf{v} \in \mathcal{S}_B} M(\mathbf{v}) (v_i \, p_j),$$

$$\frac{\partial C}{\partial c_j} = \sum_{\mathbf{v} \in \mathcal{S}_A} (1 - M(\mathbf{v})) (-p_j) + \sum_{\mathbf{v} \in \mathcal{S}_B} M(\mathbf{v}) (p_j),$$

$$\frac{\partial C}{\partial b_i} = \sum_{\mathbf{v} \in \mathcal{S}_A} (1 - M(\mathbf{v})) (b_i - v_i) + \sum_{\mathbf{v} \in \mathcal{S}_B} M(\mathbf{v}) (v_i - b_i);$$

where $p_j := P_A(h_j|\mathbf{v})$. The formulae for model B are the same with opposite sign, and with $p_j := P_B(h_j|\mathbf{v})$. Note that there is no need to assume that both RBMs have the same number of hidden units. We note that discriminative training of a single RBM was suggested in [6] and [8].

## 4 Experiments on fMRI data

### 4.1 Data and preprocessing

For our numerical experiments, we use the fMRI data from a study of recovery from stroke described in [14]. A stroke permanently damages part of the brain ( the "lesion"), resulting in loss of the corresponding function. Some function can be recovered over a period of months or years. Since the lesion is still present, the patient must have learned to used other parts of the brain to compensate. Studying stroke patients during recovery with fMRI can help determine what changes in brain function occur during recovery, and to what degree these changes correlate with degree of recovery. The study of [14] analysed mean volumes of activation over 4 regions of interest in each hemisphere. The main conclusion of that paper is that patients with good recovery have higher activations (averaged over all sessions) in the ipsilateral cerebellum.

Twelve subjects were studied at 1,2,3 and 6 months post-stroke. Due to data irregularities, we study only 9 of these subjects in this paper; Each of the four imaging sessions consisted of four continuous recording runs. During each run, the subject alternated two kinds of hand movement: tapping finger and thumb together, or wrist flexion/extension; with rest breaks in between. The movement was paced auditorily at 1Hz. During a single run, only one hand moved; during the following run, the other hand moved. Within a run, the experimental design was : (3 seconds rest, 6 seconds finger tap, 3 seconds rest, 6 seconds wrist flexion), repeated 8 times.

The fMRI images, called "volumes", are recorded every 4 seconds. The volumes are made up of 24 axial (i.e. horizontal) slices of thickness 6mm, and within each slice the pixel size is 2mm $\times$ 2mm.

The data for all 9 subjects has been co-registered and motion-corrected using the Automated Image Registration (AIR) package [16]. For computational ease, we retain only 7 horizontal fMRI slices out of an available 24 (slices 2,3,4,5,21,22,23, with 24 being the top of the head), resulting in 10499 voxels. The choice of slices is based on prior assumptions about what parts of the brain are involved in finger and wrist motion. We temporally filter the data by dividing each "active" image (finger or wrist) by the mean of the previous two rest images. We linearly scaled all of the data for each subject in such a way that the each voxel has mean 0 and variance approximately 1. So as to avoid the long transients intrinsic in fMRI imaging, we discard the first image from each movement block, and all rest images.

## 4.2 Classification tasks

We have studied two binary classification tasks. The first task is to predict whether a given fMRI volume was recorded "early" in the study, defined as the first or second recording session (1 or 2 months post-stroke) or "late" in the study, defined as the third or fourth recording session (3 or 6 months post-stroke). This task addresses our interest in the long-term changes in brain organisation and function during stroke recovery. The second task is to predict whether the volume was recorded during finger or wrist movement. Both classification tasks are complex, in the sense that each of the two classes is known to be heterogeneous. For example, in the early vs. late task, the "early" group is known to contain volumes in four sub-classes: healthy finger movement, healthy wrist movement, impaired finger movement and impaired wrist movement; and similarly for the "late" group. In addition, there are many sources of variability between volumes that are extraneous to the classification task and that are present in any fMRI study, including physiological noise, fatigue and attention.

## 4.3 Classification methods and testing procedures

We used compared four basic methods: generatively- and discriminatively- trained pairs of RBMs; logistic regression and $K$ nearest neighbours. Each method was tested on individual fMRI slices and also on the set of 7 slices described above. For the RBMs, minimization of the cost function was by gradient descent, while for logistic regression we used the conjugate gradient algorithm as implemented by Carl Rasmussen's `minimize.m`. [1]

Data for each subject is treated separately. For each subject, the data is split into three subsets: 75% training, 12.5% validation and 12.5% test. The splitting is done by first partitioning the data into 32 half-runs, each of which contains either all of the finger movement volumes or all of the wrist movement volumes for one run. One half-run contains 8 groups of 5 consecutively-recorded volumes. From each of these half-runs, one of the 8 groups was randomly chosen and assigned to the validation set, a second group was randomly assigned to the test set, and the remaining 6 were assigned to the training set. This random splitting of each half-run into training, validation and test sets was done 20 times with different random seeds, leading to 20 uncorrelated splittings. Each classification method is evaluated on each of the 20 different random splits for each subject.

Logistic regression was L1-regularized, and the value of the regularization hyperparameter was chosen by validation. The number of nearest neighbours, $K$, was also chosen by validation. The RBMs were Cauchy-regularized, which we found to be a slight improvement over L1 regularization. When testing on individual fMRI slices (which vary in size), we used 500 hidden units; while 3500 hidden units were used when testing on the larger set of 7 slices, which contained 10499 voxels. The RBM training algorithm has many variations and hyperparameters, and is very slow to run on data of this size, so rather than doing formal validation, we adjusted the algorithm informally via many experiments on data from two of the subjects, mostly using only slice 4 but sometimes using all 7 slices. These subjects were then included in the test data, so we have not observed a strict separation of training and test data for the RBM-based methods. We note that our implementation of the discriminative gradient inadvertently used a residual variance of $1/2$ instead of 1.

We also studied various blends of generative and discriminative training of a pair of RBMs, in which the cost function is a convex combination of the negative log likelihood functions,

$$L_\lambda = (1 - \lambda)L_{\text{gen}} + \lambda L_{\text{disc}}. \tag{9}$$

## 4.4 Results

The following two tables show mean misclassification errors, averaged over all 9 subjects and all 20 splittings of the data from each subject. Following each mean error is the standard deviation of the means for the 9 subjects. The first table shows mean misclassification errors early vs. late classification task:

| | log. reg. | K near. neigh. | discrim. RBM | gen. RBM |
|---|---|---|---|---|
| Slice 2 | $28.6\% \pm 7.6$ | $11.4\% \pm 6.0$ | $7.6\% \pm 2.8$ | $3.0\% \pm 2.0$ |
| Slice 3 | $27.1\% \pm 6.7$ | $12.3\% \pm 6.1$ | $10.4\% \pm 2.2$ | $2.8\% \pm 1.9$ |
| Slice 4 | $28.1\% \pm 8.2$ | $11.3\% \pm 5.4$ | $9.7\% \pm 2.3$ | $2.4\% \pm 1.8$ |
| Slice 5 | $26.3\% \pm 7.0$ | $12.6\% \pm 5.7$ | $10.0\% \pm 2.1$ | $4.2\% \pm 3.2$ |
| Slice 21 | $24.2\% \pm 8.9$ | $16.8\% \pm 7.4$ | $10.6\% \pm 4.4$ | $4.8\% \pm 3.4$ |
| Slice 22 | $23.7\% \pm 6.9$ | $15.3\% \pm 6.5$ | $9.2\% \pm 3.5$ | $3.7\% \pm 2.1$ |
| Slice 23 | $24.1\% \pm 4.7$ | $13.0\% \pm 4.7$ | $7.7\% \pm 2.6$ | $5.2\% \pm 3.7$ |
| All 7 slices | $20.1\% \pm 8.7$ | $10.0\% \pm 4.1$ | — | $0.2\% \pm 0.2$ |

In all cases, the generatively-trained RBMs outperform all of the other methods tested. We omitted discriminative training an RBM pair for the large dataset of all 7 slices together, due to the computional expense.[2]

The next table shows mean error rates for the finger vs. wrist classification task:

| | log. reg. | K near. neigh. | discrim. RBM | gen. RBM |
|---|---|---|---|---|
| Slice 4 | $17.0\% \pm 2.8$ | $11.7\% \pm 1.5$ | $9.7\% \pm 2.3$ | $21.8\% \pm 4.6$ |
| All 7 slices | $7.9\% \pm 3.0$ | $10.6\% \pm 2.3$ | $(6.9\% \pm 2.3)$ | $11.5\% \pm 1.5$ |

For this task, we did discriminatively train an RBM pair on the entire dataset, however due to the computational expense we used only 1000 hidden units instead of the the 3500 used in generative training. Experiments on one subject suggests that the results for discriminative training are not very sensitive to the number of hidden units.

Figure 1 shows the performance of several convex blends of generative and discriminative training, tested on fMRI Slice 4. Due to the computational intensity, only 5 splittings of the data were tested for each blend. Note that the for the early vs. late task, pure generative training outperforms all other blends; while for the finger vs. wrist task, pure discriminative training outperforms all other blends.

## 5   Discussion

This study shows that generative models, and in particular, Restricted Boltzmann Machines, can be very useful in discrimination tasks. It is been shown before that generative training can make use of unlabelled data to improve discriminative performance [7] [6]. The present study, like that of Ng and Jordan [11], shows that generative training can improve discriminative performance even if all data is labelled.

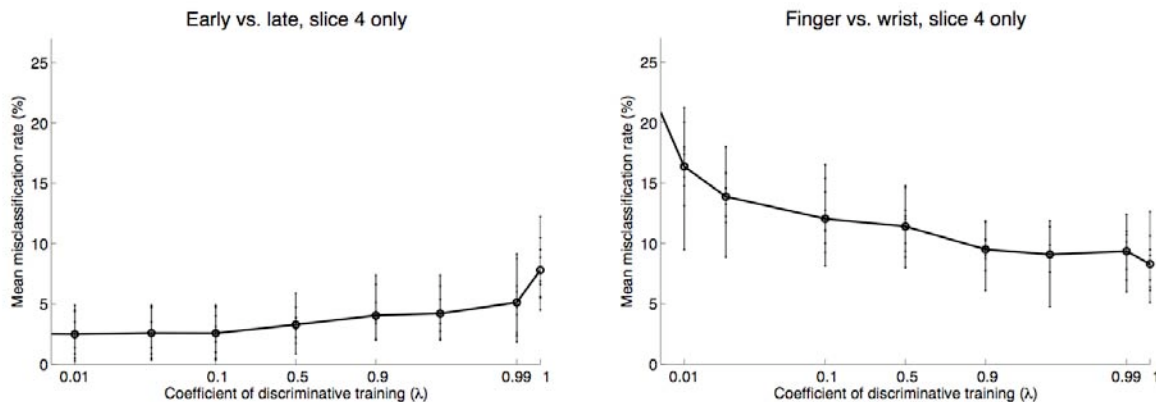

Figure 1: Misclassification rates for a combination of $(1 - \lambda)$ times generative training plus $\lambda$ times discriminative training, as in Equation (9). The $\lambda$ axis has been warped to emphasize values near 0 and 1. For each $\lambda$ value, the mean error rate across all subjects is marked with a circle. The smaller dots, joined by a vertical bar, show mean error rates for individual subjects.

We studied two methods of training a pair of RBM models: one almost entirely generative, and one discriminative. To use the terminology of Ng and Jordan, the two algorithms form a generative-discriminative pair, since they use exactly the same models of the input data and differ only in the training criterion. We found that training a pair of RBM models generatively rather than discriminatively yielded better discriminative performance for one of the two tasks studied. This is consistent with the results of Ng and Jordan, who studied the generative-discriminative pair consisting of naive Bayes and logistic regression and found that naive Bayes can outperform logistic regression. Their theoretical and experimental results suggest that generative training is more likely to be superior to discriminative training when the number of training examples is small compared to the dimension of the input data space. Since fMRI studies are in this regime, generative training looks promising for fMRI-based classification tasks.

The two tasks studied in the present work are: (i) classify fMRI volumes as having been recorded in either the earlier or later part of the study; and (ii) classify fMRI volumes as corresponding to either finger or wrist movement. We found that generative training yielded better results for the early vs. late task, while discriminative training was superior for the finger vs. wrist task.

Why does the relative performance of the two methods vary so much between the two tasks? One general observation is that generative training is trying to model many different features at once, many of which may be irrelevant to the discrimination task; whereas, by definition, discriminative models always focus on the task at hand. Thus there is a possibility for generative models to be "distracted" (from the point of view of discrimination) by rich structure in the data that is extraneous to the discrimination task. It seems reasonable that, the more structure there is in the images that is irrelevant to the discrimination task, the poorer will be the discriminative power of the generative models. We hypothesize that a lot of the complex structure in the fMRI volumes is relevant to early vs. late classification, but that most of it is irrelevant to finger vs. wrist classification. In other words, we hypothesise that the long-term changes during stroke recovery are complex and distributed throughout the brain; and that, by contrast the differences in brain activation between finger and wrist movements are relatively simple.

It is interesting to compare these results with those of [13] which shows, using the same data as the present study, that linear classification methods perform better than non-linear ones on the finger vs. wrist classification task, while for the early vs. late classification task the reverse is true.

We have evaluated blends of generative and discriminative training, as other authors have found that a combination can out-perform both pure generative and pure discriminative training [9][2]. However this did not occur in our experiments for either of the classification tasks.

From the point of view of neuroscience or medicine, this work has two ultimate aims. The first is to elucidate neural changes that occur during recovery from a stroke. This is why we chose to study the early vs. late task. This classification task may shed light on neural representations, as the regions that change over time will be those that are useful for making this discrimination. The present study identifies a specific method that is very successful at the early vs. late classification task, but does not go on to address the problem of "opening the box", i.e. shedding light on *how* the classification method works. Interpreting a set of RBM parameters is known to be more difficult than for linear models, but there are avenues available such as automatic relevance determination [10] that can indicate which voxels are most significant in the discrimination. The second aim is find general classification methods that can eventually be applied in clinical studies to classify patients as likely responders or non-responders to certain treatments on the basis of fMRI scans. We have shown that RBM-based models warrant further investigation in this context. In future work we intend to evaluate such models for their power to generalize strongly across subjects and recording sessions.

### Acknowledgments

We thank Natasa Kovacevic for co-registering and motion-correcting the fMRI data used in this study. This work was supported by the Brain Network Recovery Group through a grant from the James S. McDonnell Foundation (No. 22002082).

## Footnotes

[1]We had originally used conjugate gradient for discriminatively-trained RBMs as well but we found, late in the study, that gradient descent ran faster and gave better results. We haven't investigated this, beyond numerical verification of our gradients, but it suggests that care should be taken using conjugate gradient with very high-dimensional data.

[2]As noted earlier, we began by using conjugate gradient to train these models, and found it to be extremely slow. Now that we have switched to gradient descent, discriminative training should be of comparable speed to the generative training, which is still very computationally intensive for this dataset.

## References

[1] Y. Bengio, P. Lamblin, D. Popovici, and H. Larochelle. Greedy layer-wise training of deep networks. In *Advances in Neural Information Processing Systems 19*, pages 153– 160. MIT Press, 2007.

[2] C. M. Bishop and J. Lasserre. Generative or discriminative? getting the best of both worlds. *Bayesian Statistics*, 8:3 – 24, 2007.

[3] L. K. Hansen. Multivariate strategies in functional magnetic resonance imaging. *Brain and Language*, 102:186–191, August 2007.

[4] S. J. Hanson and Y.O. Halchenko. Brain Reading Using Full Brain Support Vector Machines for Object Recognition: There Is No "Face" Identification Area. *Neural Computation*, 20(2):486–503, 2008.

[5] G. E. Hinton. Training products of experts by minimizing contrastive divergence. *Neural Computation*, 14(8):1711–1800, 2002.

[6] G. E. Hinton. To recognize shapes, first learn to generate images. In P. Cisek, T. Drew, and J. Kalaska, editors, *Computational Neuroscience: Theoretical Insights into Brain Function*. Elsevier, 2007.

[7] G. E. Hinton and R. R. Salakhutdinov. Reducing the dimensionality of data with neural networks. *Science*, 313(5786):504 – 507, July 2006.

[8] H. Larochelle and Y. Bengio. Classification using discriminative restricted boltzmann machines. In *ICML '08: Proceedings of the 25th international conference on Machine learning*. ACM, 2008.

[9] Andrew McCallum, Chris Pal, Greg Druck, and Xuerui Wang. Multi-conditional learning: Generative/discriminative training for clustering and classification. In *To appear in AAAI '06: American Association for Artificial Intelligence National Conference on Artificial Intelligence*, 2006.

[10] R. M. Neal. *Bayesian Learning for Neural Networks*. Springer Verlag, 1996.

[11] A. Ng and M. Jordan. On discriminative vs. generative classifiers: A comparison of logistic regression and naive Bayes. *Advances in Neural Information Processing Systems 14: Proceedings of the 2002 [sic] Conference*, 2002.

[12] A.J. O'Toole, F. Jiang, H. Abdi, N. Penard, J.P. Dunlop, and M.A. Parent. Theoretical, statistical, and practical perspectives on pattern-based classification approaches to functional neuroimaging analysis. *Journal of Cognitive Neuroscience*, 19(11):1735–1752, 2007.

[13] T. Schmah, G. Yourganov, R. S. Zemel, G. E. Hinton, S. L. Small, and S. Strother. A comparison of classification methods for longitudinal fmri studies. in preparation.

[14] S. L. Small, P. Hlustik, D. C. Noll, C. Genovese, and A. Solodkin. Cerebellar hemispheric activation ipsilateral to the paretic hand correlates with functional recovery after stroke. *Brain*, 125(7):1544, 2002.

[15] M. Welling, M. Rosen-Zvi, and G. E. Hinton. Exponential family harmoniums with an application to information retrieval. In *Advances in Neural Information Processing Systems 17*. MIT Press, 2005.

[16] R. P. Woods, S. T. Grafton, C. J. Holmes, S. R. Cherry, and J. C. Mazziotta. Automated image registration: I. general methods and intrasubject, intramodality validation. *Journal of Computer Assisted Tomography*, 22:139–152, 1998.
